# Sample Size Requirements For Feedforward Neural Networks

**Michael J. Turmon**
Cornell Univ. Electrical Engineering
Ithaca, NY 14853
mjt@ee.cornell.edu

**Terrence L. Fine**
Cornell Univ. Electrical Engineering
Ithaca, NY 14853
tlfine@ee.cornell.edu

## Abstract

We estimate the number of training samples required to ensure that the performance of a neural network on its training data matches that obtained when fresh data is applied to the network. Existing estimates are higher by orders of magnitude than practice indicates. This work seeks to narrow the gap between theory and practice by transforming the problem into determining the distribution of the supremum of a random field in the space of weight vectors, which in turn is attacked by application of a recent technique called the Poisson clumping heuristic.

## 1 INTRODUCTION AND KNOWN RESULTS

We investigate the tradeoffs among *network complexity*, *training set size*, and *statistical performance* of feedforward neural networks so as to allow a reasoned choice of network architecture in the face of limited training data. Nets are functions $\eta(x; w)$, parameterized by their weight vector $w \in \mathcal{W} \subseteq R^d$, which take as input points $x \in R^k$. For classifiers, network output is restricted to $\{0, 1\}$ while for forecasting it may be any real number. The architecture of all nets under consideration is $\mathcal{N}$, whose complexity may be gauged by its Vapnik-Chervonenkis (VC) dimension $v$, the size of the largest set of inputs the architecture can classify in any desired way ('shatter'). Nets $\eta \in \mathcal{N}$ are chosen on the basis of a training set $\mathcal{T} = \{(x_i, y_i)\}_{i=1}^n$. These $n$ samples are i.i.d. according to an *unknown* probability law $P$. Performance of a network is measured by the mean-squared error

$$
\begin{aligned}
\mathcal{E}(w) &= E(\eta(x; w) - y)^2 & (1) \\
&= P(\eta(x; w) \neq y) \quad \text{(for classifiers)} & (2)
\end{aligned}
$$

and a good (perhaps not unique) net in the architecture is $w^0 = \arg\min_{w \in \mathcal{W}} \mathcal{E}(w)$. To select a net using the training set we employ the empirical error

$$\nu_T(w) = \frac{1}{n} \sum_{i=1}^{n} (\eta(x_i; w) - y_i)^2 \qquad (3)$$

sustained by $\eta(\cdot; w)$ on the training set $T$. A good choice for a classifier is then $w^* = \arg\min_{w \in \mathcal{W}} \nu_T(w)$. In these terms, the issue raised in the first sentence of the section can be restated as, "How large must $n$ be in order to ensure $\mathcal{E}(w^*) - \mathcal{E}(w^0) \leq \epsilon$ with high probability?"

For purposes of analysis we can avoid dealing directly with the stochastically chosen network $w^*$ by noting

$$\mathcal{E}(w^*) - \mathcal{E}(w^0) \leq |\nu_T(w^*) - \mathcal{E}(w^*)| + |\nu_T(w^0) - \mathcal{E}(w^0)| \leq 2 \sup_{w \in \mathcal{W}} |\nu_T(w) - \mathcal{E}(w)|$$

A bound on the last quantity is also useful in its own right.

The best-known result is in (Vapnik, 1982), introduced to the neural network community by (Baum & Haussler, 1989):

$$P(\sup_{w \in \mathcal{W}} |\nu_T(w) - \mathcal{E}(w)| \geq \epsilon) \leq 6 \frac{(2n)^v}{v!} e^{-n\epsilon^2/2} \qquad . \qquad (4)$$

This remarkable bound not only involves no unknown constant factors, but holds independent of the data distribution $P$. Analysis shows that sample sizes of about

$$n_c = (4v/\epsilon^2) \log 3/\epsilon \qquad (5)$$

are enough to force the bound below unity, after which it drops exponentially to zero. Taking $\epsilon = .1$, $v = 50$ yields $n_c = 68\,000$, which disagrees by orders of magnitude with the experience of practitioners who train such simple networks.

More recently, Talagrand (1994) has obtained the bound

$$P(\sup_{w \in \mathcal{W}} |\nu_T(w) - \mathcal{E}(w)| \geq \epsilon) \leq K_1 \left( \frac{K_2 n \epsilon^2}{v} \right)^v e^{-2n\epsilon^2}, \qquad (6)$$

yielding a sufficient condition of order $v/\epsilon^2$, but the values of $K_1$ and $K_2$ are inaccessible so the result is of no practical use.

Formulations with finer resolution near $\mathcal{E}(w) = 0$ are used. Vapnik (1982) bounds $P(\sup_{w \in \mathcal{W}} |\nu_T(w) - \mathcal{E}(w)|/\mathcal{E}(w)^{1/2} \geq \epsilon)$—note $\mathcal{E}(w)^{1/2} \approx Var(\nu_T(w))^{1/2}$ when $\mathcal{E}(w) \approx 0$—while Blumer et al. (1989) and Anthony and Biggs (1992) work with $P(\sup_{w \in \mathcal{W}} |\nu_T(w) - \mathcal{E}(w)| 1_{\{0\}}(\nu_T(w)) \geq \epsilon)$. The latter obtain the sufficient condition

$$n_c = (5.8v/\epsilon) \log 12/\epsilon \qquad (7)$$

for nets, if any, having $\nu_T(w) = 0$. If one is guaranteed to do reasonably well on the training set, a smaller order of dependence results.

Results (Turmon & Fine, 1993) for perceptrons and $P$ a Gaussian mixture imply that at least $v/280\epsilon^2$ samples are needed to force $\mathcal{E}(w^*) - \mathcal{E}(w^0) < 2\epsilon$ with high probability. (Here $w^*$ is the best linear discriminant with weights estimated from the data.) Combining with Talagrand's result, we see that the general (not assuming small $\nu_T(w)$) functional dependence is $v/\epsilon^2$.

## 2 APPLYING THE POISSON CLUMPING HEURISTIC

We adopt a new approach to the problem. For the moderately large values of $n$ we anticipate, the central limit theorem informs us that $\sqrt{n}\,[\nu_T(w) - \mathcal{E}(w)]$ has nearly the distribution of a zero-mean Gaussian random variable. It is therefore reasonable[1] to suppose that

$$P(\sup_{w \in \mathcal{W}} |\nu_T(w) - \mathcal{E}(w)| \geq \epsilon) \simeq P(\sup_{w \in \mathcal{W}} |Z(w)| \geq \epsilon\sqrt{n}) \leq 2P(\sup_{w \in \mathcal{W}} Z(w) \geq \epsilon\sqrt{n})$$

where $Z(w)$ is a Gaussian process with mean zero and covariance

$$R(w, v) = EZ(w)Z(v) = Cov\big((y - \eta(x; w))^2, (y - \eta(x; v))^2\big) \quad .$$

The problem about extrema of the original empirical process is equivalent to one about extrema of a corresponding Gaussian process.

The Poisson clumping heuristic (PCH), introduced in the remarkable (Aldous, 1989), provides a general tool for estimating such exceedance probabilities. Consider the excursions above level $b\,(= \epsilon\sqrt{n} \gg 1)$ by a stochastic process $Z(w)$. At left below, the set $\{w : Z(w) \geq b\}$ is seen as a group of "clumps" scattered in weight space $\mathcal{W}$. The PCH says that, provided $Z$ has no long-range dependence and the level $b$ is large, the centers of the clumps fall according to the points of a Poisson process on $\mathcal{W}$, and the clump shapes are independent. The vertical arrows (below right) illustrate two clump centers (points of the Poisson process); the clumps are the bars centered about the arrows.

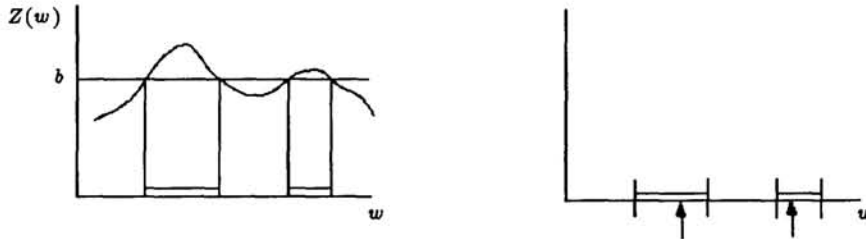

In fact, with $p_b(w) = P(Z(w) \geq b)$, $C_b(w)$ the size of a clump located at $w$, and $\lambda_b(w)$ the rate of occurrence of clump centers, the fundamental equation is

$$p_b(w) \simeq \lambda_b(w) EC_b(w). \tag{8}$$

The number of clumps in $\mathcal{W}$ is a Poisson random variable $N_b$ with parameter $\int_{\mathcal{W}} \lambda_b(w)\,dw$. The probability of a clump is $P(N_b > 0) = 1 - \exp\big(-\int_{\mathcal{W}} \lambda_b(w)\,dw\big) \simeq \int_{\mathcal{W}} \lambda_b(w)\,dw$ where the approximation holds because our goal is to operate in a regime where this probability is near zero. Letting $\bar{\Phi}(b) = P(N(0,1) > b)$ and $\sigma^2(w) = R(w, w)$, we have $p_b(w) = \bar{\Phi}(b/\sigma(w))$. The fundamental equation becomes

$$P(\sup_{w \in \mathcal{W}} Z(w) \geq b) \simeq \int_{\mathcal{W}} \frac{\bar{\Phi}(b/\sigma(w))}{EC_b(w)}\,dw \quad . \tag{9}$$

It remains only to find the mean clump size $EC_b(w)$ in terms of the network architecture and the statistics of $(x, y)$.

## 3   POISSON CLUMPING FOR SMOOTH PROCESSES

Assume $Z(w)$ has two mean-square derivatives in $w$. (If the network activation functions have two derivatives in $w$, for example, $Z(w)$ will have two almost sure derivatives.) $Z$ then has a parabolic approximation about some $w_0$ via its gradient $G = \nabla Z(w)$ and Hessian matrix $\mathbf{H} = \nabla\nabla Z(w)$ at $w_0$. Provided $Z_0 \geq b$, that is that there is a clump at $w_0$, simple computations reveal

$$C_b(w_0) \simeq \kappa_d \frac{(2(Z_0 - b) - G^T \mathbf{H}^{-1} G)^{d/2}}{|\mathbf{H}|^{1/2}} \tag{10}$$

where $\kappa_d$ is the volume of the unit ball in $R^d$ and $|\cdot|$ is the determinant. The mean clump size is the expectation of this conditioned on $Z(w_0) \geq b$.

The same argument used to show that $Z(w)$ is approximately normal shows that $G$ and $\mathbf{H}$ are approximately normal too. In fact,

$$E[\mathbf{H}|Z(w_0) = z] = \frac{z}{\sigma^2(w_0)}\Lambda(w_0)$$

$$\Lambda(w_0) = -EZ(w_0)\mathbf{H} = -\nabla_w\nabla_w R(w_0, w)|_{w=w_0}$$

so that, since $b$ (and hence $z$) is large, the second term in the numerator of (10) may be neglected. The expectation is then easily computed, resulting in

**Lemma 1 (Smooth process clump size)** *Let the network activation functions be twice continuously differentiable, and let $b \gg \sigma(w)$. Then*

$$EC_b(w) \simeq (2\pi)^{d/2}\left|\frac{\Lambda(w)}{\sigma^2(w)}\right|^{-1/2}\left(\frac{\sigma(w)}{b}\right)^d \qquad .$$

Substituting into (9) yields

$$P(\sup_{w\in W} Z(w) \geq b) \simeq (2\pi)^{-\frac{d+1}{2}} \int_W \left|\frac{\Lambda(w)}{\sigma^2(w)}\right|^{1/2}\left(\frac{b}{\sigma(w)}\right)^{d-1} e^{-b^2/2\sigma^2(w)}\, dw, \tag{11}$$

where use of the asymptotic expansion $\bar{\Phi}(z) \simeq (z\sqrt{2\pi})^{-1}\exp(-z^2/2)$ is justified since $(\forall w)b \gg \sigma(w)$ is necessary to have the individual $P(Z(w) \geq b)$ low—let alone the supremum. To go farther, we need information about the variance $\sigma^2(w)$ of $(y - \eta(x; w))^2$. In general this must come from the problem at hand, but suppose for example the process has a unique variance maximum $\bar{\sigma}^2$ at $\bar{w}$. Then, since the level $b$ is large, we can use Laplace's method to approximate the $d$-dimensional integral.

Laplace's method finds asymptotic expansions for integrals

$$\int_W g(w)\exp(-f(w)^2/2)\, dw$$

when $f(w)$ is $C^2$ with a unique positive minimum at $w_0$ in the interior of $W \subseteq R^d$, and $g(w)$ is positive and continuous. Suppose $f(w_0) \gg 1$ so that the exponential factor is decreasing much faster than the slowly varying $g$. Expanding $f$ to second order about $w_0$, substituting into the exponential, and performing the integral shows that

$$\int_W g(w)\exp(-f(w)^2/2)\, dw \simeq (2\pi)^{d/2}|f(w_0)K|^{-1/2}g(w_0)\exp(-f(w_0)^2/2)$$

where $K = \nabla\nabla f(w)|_{w_0}$, the Hessian of $f$. See (Wong, 1989) for a proof. Applying this to (11) and using the asymptotic expansion for $\bar{\Phi}$ in reverse yields

**Theorem 1** *Let the network activation functions be twice continuously differentiable. Let the variance have a unique maximum $\bar{\sigma}$ at $\bar{w}$ in the interior of $\mathcal{W}$ and the level $b \gg \bar{\sigma}$. Then the PCH estimate of exceedance probability is given by*

$$P(\sup_{w \in \mathcal{W}} Z(w) \geq b) \simeq \frac{|\Lambda(\bar{w})|^{1/2}}{|\Lambda(\bar{w}) - \Gamma(\bar{w})|^{1/2}} \bar{\Phi}(b/\bar{\sigma}) \tag{12}$$

*where $\Gamma(\bar{w}) = \nabla_w \nabla_v R(w,v)|_{w=v=\bar{w}}$. Furthermore, $\Lambda - \Gamma$ is positive-definite at $\bar{w}$; it is $-1/2$ the Hessian of $\sigma^2(w)$. The leading constant thus strictly exceeds unity.*

The above probability is just $P(Z(\bar{w}) \geq b)$ multiplied by a factor accounting for the other networks in the supremum. Letting $b = \epsilon\sqrt{n}$ reveals

$$n_c = \frac{\bar{\sigma}^2 \log(|\Lambda(\bar{w})|/|\Lambda(\bar{w}) - \Gamma(\bar{w})|)}{\epsilon^2} \tag{13}$$

samples force $P(\sup_w |\nu_T(w) - \mathcal{E}(w)| \geq \epsilon)$ below unity. If the variance maximum is not unique but occurs over a $\bar{d}$-dimensional set within $\mathcal{W}$, the sample size estimate becomes proportional to $\bar{\sigma}^2 \bar{d}/\epsilon^2$. With $\bar{d}$ playing the role of VC dimension $v$, this is similar to Vapnik's bound although we retain dependence on $P$ and $\mathcal{N}$.

The above probability is determined by behavior near the maximum-variance point, which for example in classification is where $\mathcal{E}(w) = 1/2$. Such nets are uninteresting as classifiers, and certainly it is undesirable for them to dominate the entire probability. This problem is avoided by replacing $Z(w)$ with $Z(w)/\sigma(w)$, which additionally allows a finer resolution where $\mathcal{E}(w)$ nears zero. Indeed, for classification, if $n$ is such that with high probability

$$\sup_{w \in \mathcal{W}} \frac{|\nu_T(w) - \mathcal{E}(w)|}{\sigma(w)} = \sup_{w \in \mathcal{W}} \frac{|\nu_T(w) - \mathcal{E}(w)|}{\sqrt{\mathcal{E}(w)(1 - \mathcal{E}(w))}} < \epsilon \quad , \tag{14}$$

then $\nu_T(w^*) = 0 \Rightarrow \mathcal{E}(w^*) < \epsilon^2(1 + \epsilon^2)^{-1} \simeq \epsilon^2 \ll \epsilon$. Near $\nu_T(w^*) = 0$, condition (14)/ is much more powerful than the corresponding unnormalized one. Sample size estimates using this setup give results having a functional form similar to (7).

## 4 ANOTHER MEANS OF COMPUTING CLUMP SIZE

Conditional on there being a clump center at $w$, the upper bound

$$C_b(w) \leq D_b(w) \equiv \int_{\mathcal{W}} 1_{[0,\infty)}(Z(w') - b)\, dw' \tag{15}$$

is evidently valid: the volume of the clump at $w$ is no larger than the total volume of all clumps. (The right hand side is indeed a function of $w$ because we condition on occurrence of a clump center at $w$.) The bound is an overestimate when the number $N_b$ of clumps exceeds one, but recall that we are in a regime where $b$ (equivalently $n$) is large enough so that $P(N_b > 1)/P(N_b = 1) \simeq \int_{\mathcal{W}} \lambda_b(w)\, dw \ll 1$. Thus error in (15) due to this source is negligible. To compute its mean, we approximate

$$ED_b(w) = \int_{\mathcal{W}} P(Z(w') \geq b | w \text{ a clump center})\, dw'$$

$$\simeq \int_{\mathcal{W}} P(Z(w') \geq b | Z(w) \geq b)\, dw' \qquad . \tag{16}$$

The point is that occurrence of a clump center at $w_0$ is a smaller class of events than merely $Z(w_0) \geq b$: the latter can arise from a clump center at a nearby $w \in \mathcal{W}$ capturing $w_0$. Since $Z(w)$ and $Z(w')$ are jointly normal, abbreviate $\sigma = \sigma(w)$, $\sigma' = \sigma(w')$, $\rho = \rho(w,w') = R(w,w')/(\sigma\sigma')$, and let

$$\zeta = \zeta(w,w') \;=\; (\sigma/\sigma')\frac{1 - \rho\sigma'/\sigma}{\sqrt{1-\rho^2}} \tag{17}$$

$$= \;\; ((1-\rho)/(1+\rho))^{1/2} \quad \text{(constant variance case)} \qquad . \tag{18}$$

Evaluating the conditional probabilities of (16) presents no problem, and we obtain

**Lemma 2 (Clump size estimate)** *For $b \gg \sigma$ the mean clump size is*

$$EC_b(w) \simeq ED_b(w) \simeq \int_{\mathcal{W}} \bar{\Phi}((b/\sigma)\zeta)\, dw' \qquad . \tag{19}$$

**Remark 1.** This integral will be used in (9) to find

$$P(\sup_w Z(w) > b) \simeq \int_{\mathcal{W}} \frac{\bar{\Phi}(b/\sigma)}{\int_{\mathcal{W}} \bar{\Phi}((b/\sigma)\zeta)\, dw'}\, dw \qquad . \tag{20}$$

Since $b$ is large, the main contribution to the outer integral occurs for $w$ near a variance maximum, i.e. for $\sigma'/\sigma \leq 1$. If the variance is constant then all $w \in \mathcal{W}$ contribute. In either case $\zeta$ is nonnegative. By lemma 1 we expect (19) to be, as a function of $b$, of the form $(\text{const } \sigma/b)^p$ for, say, $p = d$. In particular, we do not anticipate the exponentially small clump sizes resulting if $(\forall w')\zeta(w,w') \geq M \gg 0$. Therefore $\zeta$ should approach zero over some range of $w'$, which happens only when $\rho \approx 1$, that is, for $w'$ near $w$. The behavior of $\rho(w,w')$ for $w' \approx w$ is the key to finding the clump size.

**Remark 2.** There is a simple interpretation of the clump size; it represents the volume of $w' \in \mathcal{W}$ for which $Z(w')$ is highly correlated with $Z(w)$. The exceedance probability is a sum of the point exceedance probabilities (the numerator of (20)), each weighted according to how many other points are correlated with it. In effect, the space $\mathcal{W}$ is partitioned into regions that tend to "have exceedances together," with a large clump size $EC_b(w)$ indicating a large region. The overall probability can be viewed as a sum over all these regions of the corresponding point exceedance probability. This has a similarity to the Vapnik argument which lumps networks together according to their $n^v/v!$ possible actions on $n$ items in the training set. In this sense the mean clump size is a fundamental quantity expressing the ability of an architecture to generalize.

## 5   EMPIRICAL ESTIMATES OF CLUMP SIZE

The clump size estimate of lemma 2 is useful in its own right if one has information about the covariance of $Z$. Other known techniques of finding $EC_b(w)$ exploit special features of the process at hand (e.g. smoothness or similarity to other well-studied processes); the above expression is valid for any covariance structure. In

this section we show how one may *estimate* the clump size using the training set, and thus obtain probability approximations in the absence of analytical information about the unknown $P$ and the potentially complex network architecture $\mathcal{N}$.

Here is a practical way to approximate the integral giving $ED_b(w)$. For $\gamma < 1$ define a set of significant $w'$

$$S_\gamma(w) = \{w' \in \mathcal{W} : \zeta(w, w') \le \gamma\} \qquad V_\gamma(w) = \mathrm{vol}(S_\gamma(w)) \qquad ; \qquad (21)$$

then monotonicity of $\bar{\Phi}$ yields $ED_b(w) \ge \int_{S_\gamma} \bar{\Phi}((b/\sigma)\zeta)\,dw' \ge V_\gamma(w)\,\bar{\Phi}((b/\sigma)\gamma)$. This apparently crude lower bound for $\bar{\Phi}$ is accurate enough near the origin to give satisfactory results in the cases we have studied. For example, we can characterize the covariance $R(w, w')$ of the smooth process of lemma 1 and thus find its $\zeta$ function. The bound above is then easily calculated and differs by only small constant factors from the clump size in the lemma.

The lower bound for $ED_b(w)$ yields the upper bound

$$P(\sup_w Z(w) \ge b) \le \int_\mathcal{W} \frac{\bar{\Phi}(b/\sigma)}{V_\gamma(w)\,\bar{\Phi}((b/\sigma)\gamma)}\,dw \quad . \qquad (22)$$

We call $V_\gamma(w)$ the *correlation volume*, as it represents those weight vectors $w'$ whose errors $Z(w')$ are highly correlated with $Z(w)$; one simple way to estimate the correlation volume is as follows. Select a weight $w'$ and using the training set compute

$$(y_1 - \eta(x_1; w))^2, \ldots, (y_n - \eta(x_n; w))^2 \ \& \ (y_1 - \eta(x_1; w'))^2, \ldots, (y_n - \eta(x_n; w'))^2 \ .$$

It is then easy to estimate $\sigma^2$, $\sigma'^2$, and $\rho$, and finally $\zeta(w, w')$, which is compared to the chosen $\gamma$ to decide if $w' \in S_\gamma(w)$.

The difficulty is that for large $d$, $S_\gamma(w)$ is far smaller than any approximately-enclosing set. Simple Monte Carlo sampling and even importance sampling methods fail to estimate the volume of such high-dimensional convex bodies because so few hits occur in probing the space (Lovász, 1991). The simplest way to concentrate the search is to let $w' = w$ except in one coordinate and probe along each coordinate axis. The correlation volume is approximated as the product of the one-dimensional measurements.

Simulation studies of the above approach have been performed for a perceptron architecture in input uniform over $[-1, 1]^d$. The integral (22) is computed by Monte Carlo sampling, and based on a training set of size $100d$, $V_\gamma(w)$ is computed at each point via the above method. The result is that an estimated sample size of $5.4d/\epsilon^2$ is enough to ensure (14) with high probability. For nets, if any, having $\nu_T(w) = 0$, sample sizes larger than $5.4d/\epsilon$ will ensure reliable generalization, which compares favorably with (7).

## 6 SUMMARY AND CONCLUSIONS

To find realistic estimates of sample size we transform the original problem into one of finding the distribution of the supremum of a derived Gaussian random field, which is defined over the weight space of the network architecture. The latter problem is amenable to solution via the Poisson clumping heuristic. In terms of the PCH the question becomes one of estimating the mean clump size, that

is, the typical volume of an excursion above a given level by the random field. In the "smooth" case we directly find the clump volume and obtain estimates of sample size that are (correctly) of order $v/\epsilon^2$. The leading constant, while explicit, depends on properties of the architecture and the data—which has the advantage of being tailored to the given problem but the potential disadvantage of our having to compute them.

We also obtain a useful estimate for the clump size of a general process in terms of the correlation volume $V_\gamma(w)$. For normalized error, (22) becomes approximately

$$P\left(\sup_{w \in \mathcal{W}} \frac{\nu_T(w) - \mathcal{E}(w)}{\sigma(w)} \geq \epsilon\right) \approx E\left[\frac{\text{vol}(\mathcal{W})}{V_\gamma(w)}\right] e^{-(1-\gamma^2)n\epsilon^2/2}$$

where the expectation is taken with respect to a uniform distribution on $\mathcal{W}$. The probability of reliable generalization is roughly given by an exponentially decreasing factor (the exceedance probability for a single point) times a number representing degrees of freedom. The latter is the mean size of an equivalence class of "similarly-acting" networks. The parallel with the Vapnik approach, in which a worst-case exceedance probability is multiplied by a growth function bounding the number of classes of networks in $\mathcal{N}$ that can act differently on $n$ pieces of data, is striking. In this fashion the correlation volume is an analog of the VC dimension, but one that depends on the interaction of the data and the architecture.

Lastly, we have proposed practical methods of estimating the correlation volume empirically from the training data. Initial simulation studies based on a perceptron with input uniform on a region in $R^d$ show that these approximations can indeed yield informative estimates of sample complexity.

## Footnotes

[1]See ch. 7 of (Pollard, 1984) for treatment of some technical details in this limit.

## References

Aldous, D. 1989. *Probability Approximations via the Poisson Clumping Heuristic.* Springer.

Anthony, M., & Biggs, N. 1992. *Computational Learning Theory.* Cambridge Univ.

Baum, E., & Haussler, D. 1989. What size net gives valid generalization? *Pages 81–90 of:* Touretzky, D. S. (ed), *NIPS 1.*

Blumer, A., Ehrenfeucht, A., Haussler, D., & Warmuth, M. K. 1989. Learnability and the Vapnik-Chervonenkis dimension. *Jour. Assoc. Comp. Mach.*, **36**, 929–965.

Lovász, L. 1991. Geometric Algorithms and Algorithmic Geometry. *In: Proc. Internat. Congr. Mathematicians.* The Math. Soc. of Japan.

Pollard, D. 1984. *Convergence of Stochastic Processes.* Springer.

Talagrand, M. 1994. Sharper bounds for Gaussian and empirical processes. *Ann. Probab.*, **22**, 28–76.

Turmon, M. J., & Fine, T. L. 1993. Sample Size Requirements of Feedforward Neural Network Classifiers. *In: IEEE 1993 Intern. Sympos. Inform. Theory.*

Vapnik, V. 1982. *Estimation of Dependences Based on Empirical Data.* Springer.

Wong, R. 1989. *Asymptotic Approximations of Integrals.* Academic.